# A MASSIVELY PARALLEL SELF-TUNING CONTEXT-FREE PARSER[1]

Eugene Santos Jr.
Department of Computer Science
Brown University
Box 1910, Providence, RI 02912
esj@cs.brown.edu

## ABSTRACT

The Parsing and Learning System(PALS) is a massively parallel self-tuning context-free parser. It is capable of parsing sentences of unbounded length mainly due to its parse-tree representation scheme. The system is capable of improving its parsing performance through the presentation of training examples.

## INTRODUCTION

Recent PDP research[Rumelhart et al., 1986; Feldman and Ballard, 1982; Lippmann, 1987] involving natural language processing[Fanty, 1988; Selman, 1985; Waltz and Pollack, 1985] have unrealistically restricted sentences to a fixed length. A solution to this problem was presented in the system CONPARSE[Charniak and Santos, 1987]. A parse-tree representation scheme was utilized which allowed for processing sentences of any length. Although successful as a parser, it's achitecture was strictly hand-constructed with no learning of any form. Also, standard learning schemes were not applicable since it differed from all the popular architectures, in particular, connectionist ones.

In this paper, we present the Parsing and Learning System(PALS) which attempts to integrate a learning scheme into CONPARSE. It basically allows CONPARSE to modify and improve its parsing capability.

## REPRESENTATION OF PARSE TREE

A parse-tree is represented by a matrix where the bottom row consists of the leaves of the tree in left-to-right order and the entries in each column above each leaf correspond to the nodes in the path from leaf to root. For example, looking at the simple parse-tree for the sentence "noun verb noun", the column entries for verb would be verb, VP, and S. (see Figure 1) (As in previous work, PALS takes part-of-speech as input, not words.)

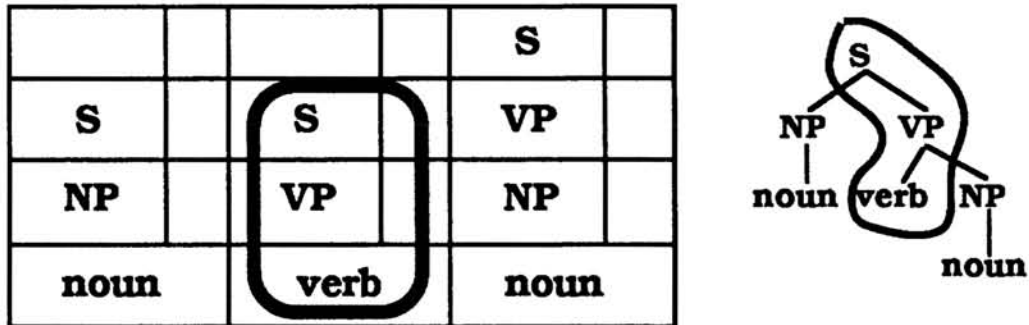

**Figure 1.** Parse tree as represented by a collection of columns in the matrix.

In addition to the columns of nonterminals, we introduce the binder entries as a means of easily determining whether two identical nonterminals in adjacent columns represent the same nonterminal in a parse tree (see Figure 2).

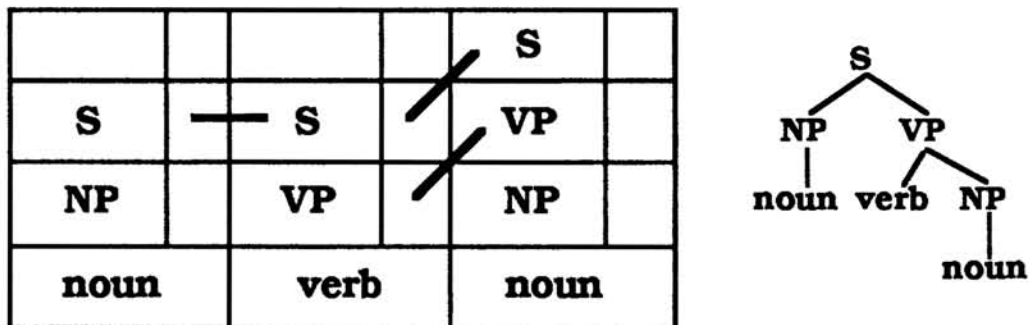

**Figure 2.** Parse tree as represented by a collection of columns in the matrix plus binders.

To distributively represent the matrix, each entry denotes a collection of labeled computational units. The value of the entry is taken to be the label of the unit with the largest value.

A nonterminal entry has units which are labeled with the nonterminals of a language plus a special label "blank". When the "blank" unit is largest, this indicates that the entry plays no part in representing the current parse tree.

A binder entry has units which are labeled from 1 to the number of rows in the matrix. A unit labeled k then denotes the binding of the nonterminal entry on its immediate left to the nonterminal entry in the kth row on its right. To indicate that no binding exists, we use a special unit label "e" called an edge.

In general, it is easiest to view an entry as a vector of real numbers where each vector component denotes some symbol. (For more information see [Charniak and Santos, 1987].)

In the current implementation of PALS, entry unit values range from 0 to 1. The ideal value for entry units is thus 1 for the largest entry unit and 0 for all remaining entry units. We essentially have "1" being "yes" and "0" being no.

## LANGUAGE RULES

In order to determine the values of the computational units mentioned in the previous section, we apply a set of *language rules*. Each compuatational unit will be determined by some subset of these rules.

Each language rule is represented by a single node, called a *rule node*. A rule node takes its input from several computational units and outputs to a single computational unit.

The output of each rule node is also modified by a non-negative value called a *rule-weight*. This weight represents the applicability of a language rule to the language we are attempting to parse (see PARSING). In the current implementation of PALS, rule-weight values range from 0 to 1 being similar to probabilities.

Basically, a rule node attempts to express some rule of grammar. As with CONPARSE, PALS breaks context-free grammars into several subrules. For example, as part of S --> NP VP, PALS would have a rule stating that an NP entry would like to have an S immediately above it in the same column. Our rule for this grammar rule will then take as input the entry's computational unit labeled NP and output to the unit labeled S in the entry immediately above(see Figure 3).

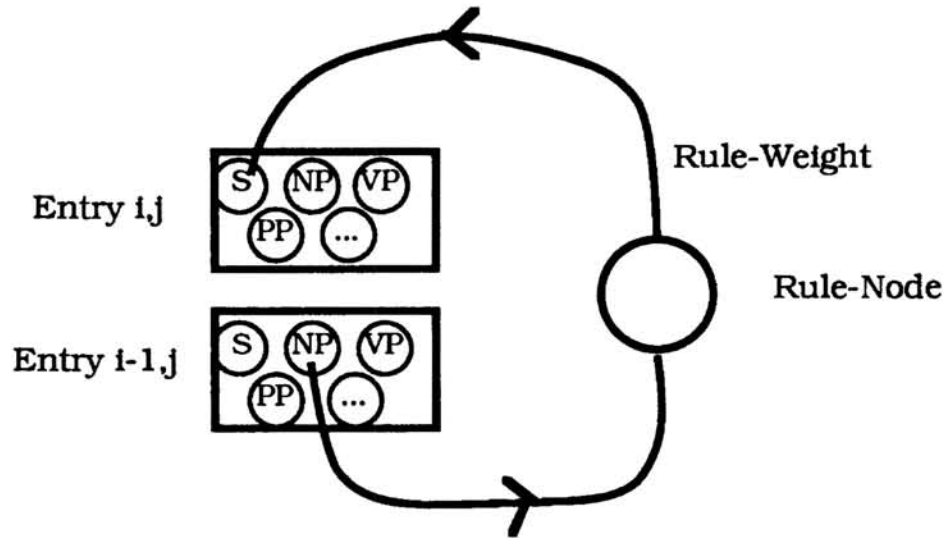

**Figure 3.** A rule node for S above NP.

As a more complex example, if an entry is a NP, the NP does not continue right, i.e., has an edge, and above is an S that continues to the right, then below the second S is a VP.

In general, to determine a unit's value, we take all the rule nodes and combine their influences. This will be much clearer when we discuss parsing in the next section.

## PARSING

Since we are dealing with a PDP-type architecture, the size of our matrix is fixed ahead of time. However, the way we use the matrix representation scheme allows us to handle sentences of unbounded length as we shall see.

The system parses a sentence by taking the first word and placing it in the lower rightmost entry; it then attempts to construct the column above the word by using its rule nodes. After this processing, the system shifts the first word and its column left and inserts the second word. Now both words are processed simultaneously. This shifting and processing continues until the last word is shifted through the matrix (see Figure 4). Since sentence lengths may exceed the size of the matrix, we are only processing a portion at a time, creating partial parse-trees. The complete parse-tree is the combination of these partial ones.

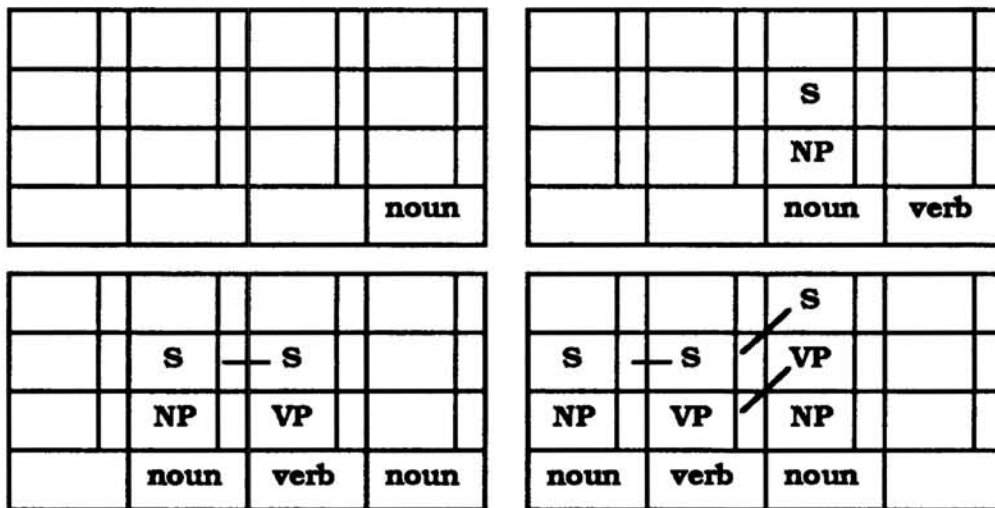

**Figure 4.** Parsing of noun verb noun.

Basically, the system builds the tree in a bottom-up fashion. However, it can also build left-right, right-left, and top-down since columns may be of differing height. In general, columns on the left in the matrix will be more complete and hence possibly higher than those on the right.

## LEARNING

The goal of PALS is to learn how to parse a given language. Given a system consisting of a matrix with a set of language rules, we learn parsing by determining how to apply each language rule.

In general, when a language rule is inconsistent with the language we are learning to parse, its corresponding rule-weight drops to zero, essentially disconnecting the rule. When a language rule is consistent, its rule-weight then approaches one.

In PALS, we learn how to parse a sentence by using training examples. The teacher/trainer gives to the system the complete parse tree of the sentence to be learned.

Because of the restrictions imposed by our matrix, we may be unable to fully represent the complete parse tree given by the teacher. To learn how to parse the sentence, we can only utilize a portion of the complete parse tree at any one time.

Given a complete parse tree, the system simply breaks it up into manageable chunks we call *snapshots*. Snapshots are matrices which contain a portion of the complete parse tree.

Given this sequence of snapshots, we present them to the system in a fashion similar to parsing. The only difference is that we clamped the

snapshot to the system matrix while it fires its rule nodes. From this, we can easily determine whether a rule node has incorrectly fired or not by seeing if it fired consistently with given snapshot. We punish and reward accordingly.

As the system is trained more and more, our rule-weights contain more and more information. We would like the rule-weights of those rules used frequently during training to not change as much as those not frequently used. This serves to stabilize our knowledge. It also prevents the system from being totally corrupted when presented with an incorrect training example.

As in traditional methods, we find the new rule-weights by minimizing some function which gives us our desired learning. The function which captures this learning method is

$$\Sigma_{i,j} \{c_{i,j} ( \alpha_{i,j} - \beta_{i,j} )^2 + [ \delta_{i,j} \beta_{i,j} + ( 1 - \delta_{i,j} ) ( 1 - \beta_{i,j} ) ]^2 r_{i,j}^2\}$$

where i are the unit labels for some matrix entry, j are the language rules associated with units i, $\alpha_{i,j}$ are the old rule-weights, $\beta_{i,j}$ are the new rule-weights, $c_{i,j}$ is the knowledge preservation coefficient which is a function of the frequency that language rule j for entry unit i has been fired during learning, $r_{i,j}$ is the unmodified rule output using snapshot as input, and $\delta_{i,j}$ is the measure of the correctness of language rule j for unit i.

## RESULTS

In the current implementation of PALS, we utilize a 7x6 matrix and an average of fifty language rules per entry unit to parse English.

Obviously, our set of language rules will determine what we can and cannot learn. Currently, the system is able to learn and parse a modest subset of the English language. It can parse sentences with moderate sentence embedding and phrase attachment from the following grammar:

| | |
|---|---|
| SM | --> S per |
| S | --> NP VP |
| N P | --> (det) (adj)* noun (PP)* (WHCL) (INFP1) |
| NP | --> INFP2 |
| PP | --> prep NP |
| WHCL | --> that S/NP |
| S/NP | --> VP |
| S/NP | --> NP VP/NP |

```
INFP1        --> (NP) INF1
INF1         --> to (adv) VP/NP
VP           --> (aux) trverb NP (PP)*
VP           --> (aux) intrverb
VP           --> (aux) copverb NP
VP           --> (aux) copverb PP
VP           --> (aux) copverb adj
VP/NP        --> (aux) trverb (PP)*
INFP2        --> (NP) INF2
INF2         --> to (adv) VP
```

We have found that sentences only require approximately two trainings.

We have also found that by adding more consistent language rules, the system improved by actually generating parse trees which were more "certain" than previously generated ones. In other words, the values of the entry units in the final parse tree were much closer to the ideal.

When we added inconsistent language rules, the system degraded. However, with slightly more training, the system was back to normal. It actually had to first eliminate the inconsistent rules before being able to apply the consistent ones.

Finally, we attempted to train the system with incorrect training examples after being trained with correct ones. We found that even though the system degraded slightly, previous learning was not completely lost. This was basically due to the stability employed during learning.

## CONCLUSIONS

We have presented a system capable of parsing and learning how to parse.

The system parses by creating a sequence of partial parse trees and then combining them into a parse tree. It also places no limit on sentence length.

Given a system consisting of a matrix and an associated set of language rules, we attempt to learn how to parse the language described by the complete parse tree supplied by a teacher/trainer. The same set of language rules may also be able to learn a different language. Depending on the diversity of the language rules, it may also learn both simultaneously, i.e., parse both languages. (A simple example is two languages with distinct terminals and nonterminals.)

The system learns by being presented complete parse-trees and adding to its knowledge by modifying its rule-weights.

The system requires a small number of trainings per training example. Also, incorrect training examples do not totally corrupt the system.

## PROBLEMS AND FUTURE RESEARCH

Eventhough the PALS system has managed to integrate learning, there are still some problems. First, as in the CONPARSE system, we can only handle moderately embedded sentences. Second, the system is very positional. Something that is learned in one portion of the matrix is not generalized to other portions.

There is no rule aquisition in the PALS system. Currently, all rules are assumed to be built-in to the system. PALS's ability to suppress incorrect rules would entail rule learning if the possible set of language rules was very tightly constrained so that, in effect, all rules could be tried. Some linguists have suggested quite limited schemes but if any would work with PALS is not known.

**REFERENCES**

Rumelhart, D. et al., *Parallel Distributed Processing: Explorations in the Microstructures of Cognition, Volume 1*, The MIT Press (1986).

Charniak, E. and Santos, E., "A connectionist context-free parser which is not context-free, but then it is not really connectionist either," *The Ninth Annual Conference of the Cognitive Science Society* pp. 70-77 (1987).

Fanty, M., "Learning in Structured Connectionist Networks," TR 252, University of Rochester Computer Science Department (1988).

Selman, Bart, "Rule-based processing in a connectionist system for natural language understanding," Technical Report CSRI-168, Computer Systems Research Institute, University of Toronto (1985).

Waltz, D. and Pollack, J., "Massively parallel parsing: a strongly interactive model of natural language interpretation," *Cognitive Science* **9** pp. 51-74 (1985).

Feldman, J.A. and Ballard, D.H., "Connectionist models and their properties," *Cognitive Science* **6** pp. 205-254 (1982).

Lippmann, R., "An introduction to computing with neural nets," *IEEE ASSP Magazine* pp. 4-22 (April 1987).

## Footnotes

[1]This research was supported in part by the Office of Naval Research under contract N00014-79-C-0592, the National Science Foundation under contracts IST-8416034 and IST-8515005, and by the Defense Advanced Research Projects Agency under ARPA Order No. 4786.
